# Modeling Midazolam's Effect on the Hippocampus and Recognition Memory

**Kenneth J. Malmberg**
Department of Psychology
Indiana University
Bloomington, IN 47405
*malmberg@indiana.edu*

**René Zeelenberg**
Department of Psychology
Indiana University
Bloomington, IN 47405
*rzeelenb@indiana.edu*

**Richard M. Shiffrin**
Departments of Cognitive Science and Psychology
Indiana University
Bloomington, IN 47405
*shiffrin@indiana.edu*

## Abstract

The benzodiazepine Midazolam causes dense, but temporary, anterograde amnesia, similar to that produced by hippocampal damage. Does the action of Midazolam on the hippocampus cause less storage, or less accurate storage, of information in episodic long-term memory? We used a simple variant of the REM model [18] to fit data collected by Hirshman, Fisher, Henthorn, Arndt, and Passannante [9] on the effects of Midazolam, study time, and normative word-frequency on both yes-no and remember-know recognition memory. That a simple strength model fit well was contrary to the expectations of Hirshman et al. More important, within the Bayesian based REM modeling framework, the data were consistent with the view that Midazolam causes less accurate storage, rather than less storage, of information in episodic memory.

## 1 Introduction

Damage to the hippocampus (and nearby regions), often caused by lesions, leaves normal cognitive function intact in the short term, including long-term memory retrieval, but prevents learning of new information. We have found a way to begin to distinguish two alternative accounts for this learning deficit: Does damage cause less storage, or less accurate storage, of information in long-term episodic memory? We addressed this question by using the REM model of recognition memory [18] to fit data collected by Hirshman and colleagues [9], who tested recognition memory in normal participants given either saline (control group) or Midazolam, a benzodiazepine that temporarily causes anterograde amnesia with effects that generally mimic those found after hippocampal damage.

## 2 Empirical findings

The participants in Hirshman et al. [9] studied lists of words that varied in normative word frequency (i.e., low-frequency vs. high-frequency) and the amount of time allocated for study (either not studied, or studied for 500, 1200, or 2500 ms per word). These variables are known to have a robust effect on recognition memory in normal populations: Low-frequency (LF) words are better recognized than high-frequency (HF) words, and an increase in study time improves recognition performance. In addition, the probability of responding 'old' to studied words (termed hit rate, or HR) is higher for LF words than for HF words, and the probability of responding 'old' to unstudied words (termed false alarm rate, or FAR) is lower for LF words than for HF words. This pattern of data is commonly known as a "mirror effect" [7].

In Hirshman et al. [9], participants received either saline or Midazolam and then studied a list of words. After a delay of about an hour they were shown studied words ('old') and unstudied words ('new'), and asked to give old-new recognition and remember/know judgments. The HR and FAR findings are depicted in Figure 1 as the large circles (filled for LF test words and unfilled for HF test words). The results from the saline condition, given in the left panel, replicate the standard effects in the literature: In the figure, the points labeled with zero study time give FARs (for new test items), and the other points give HRs (for old test items). Thus we see that the saline group exhibits better performance for LF words and a mirror effect: For LF words, FARs are lower and HRs are higher. The Midazolam group of course gave lower performance (see right panel). More critically, the pattern of results differs from that for the saline group: The mirror effect was lost. LF words produced both lower FARs and lower HRs.

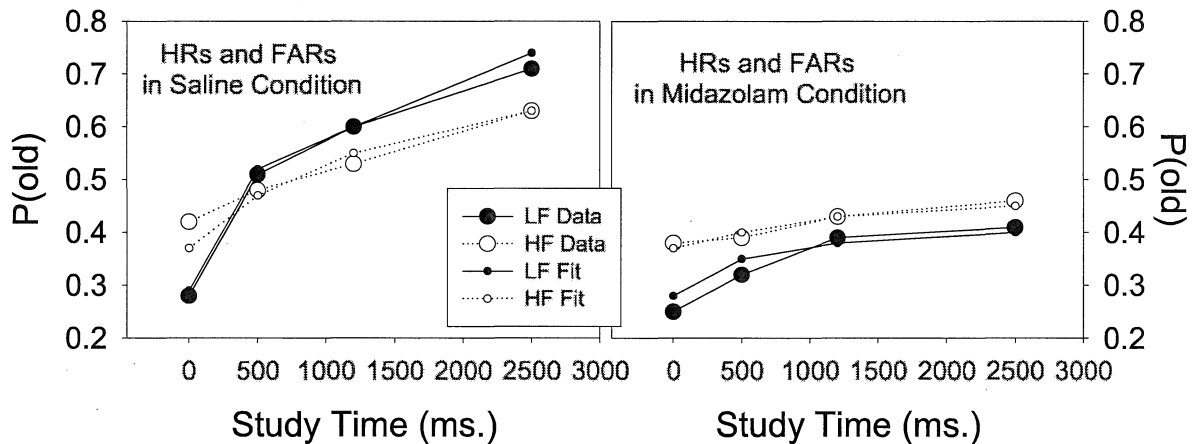

**Figure 1.** Yes-no recognition data from Hirshman et al. and predictions of a REM model. Zero ms study time refers to 'new' items so the data gives the false-alarm rate (FAR). Data shown for non-zero study times give hit rates (HR). Only the REM parameter $c$ varies between the saline and midazolam conditions. The fits are based on 300 Monte Carlo simulations using $g_{LF} = .325$, $g = .40$, $g_{HF} = .45$, $w = 16$, $t_0 = 4$, $a = .8$, $u^* = .025$, $c_{Sal} = .77$, $c_{Mid} = .25$, $Crit_{O/N} = .92$. LF = low-frequency words and HF = high-frequency words.

The participants also indicated whether their "old" judgments were made on the basis of "remembering" the study event or on the basis of "knowing" the word was studied even though they could not explicitly remember the study event [5]. Data are shown in Figure 2. Of greatest interest for present purposes, "know" and "remember" responses were differently affected by the word frequency and the drug manipulations. In the Midazolam condition, the conditional probability of a "know" judgment (given an "old" response) was consistently higher than that of a "remember" judgment (for both HF and LF words). Moreover, these probabilities were hardly affected by study time. A different pattern was obtained in the Saline condition. For HF words, the conditional probability of a "know" judgment was higher than that of a "remember" judgment, but the difference decreased with study time. Finally, for LF words, the conditional probability of a "know" judgment was higher than that of a "remember" judgment for nonstudied foils, but for studied targets the conditional probability of a "remember" judgment was higher than that of a "know" judgment. The recognition and remember/know results were interpreted by Hirshman et al. [9] to require a dual process account; in particular, the authors argued against "memory strength" accounts [4, 6, 11]. Although not the main message of this note, it will be of some interest to memory theorists to note that our present results shows this conclusion to be incorrect.

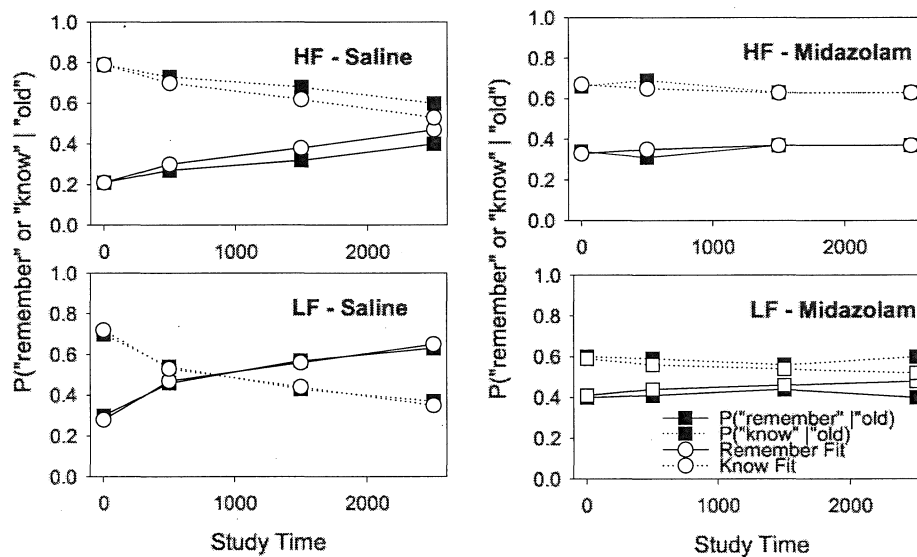

**Figure 2.** Remember/know data from Hirshman et al. and predictions of a REM model. The parameter values are those listed in the caption for Figure 1, plus there are two remember-know criterion: For the saline group, $\text{Crit}_{R/K} = 1.52$; for the midazolam group, $\text{Crit}_{R/K} = 1.30$.

## 3 A REM model for recognition and remember/know judgments

A common way to conceive of recognition memory is to posit that memory is probed with the test item, and the recognition decision is based on a continuous random variable that is often conceptualized as the resultant strength, intensity, or familiarity [6]. If the familiarity exceeds a subjective criterion, then the subject responds "old". Otherwise, a "new" response is made [8].

A subclass of this type of model accounts for the word-frequency mirror effect by assuming that there exist four underlying distributions of familiarity

values, such that the means of these distributions are arranged along a familiarity scale in the following manner: $\mu$(LF-new) < $\mu$(HF-new) < $\mu$(HF-old) < $\mu$(LF-old). The left side of Figure 3 displays this relation graphically. A model of this type can predict the recognition findings of Hirshman et al. (in press) if the effect of Midazolam is to rearrange the underlying distributions on the familiarity scale such that $\mu$(LF-old) < $\mu$(HF-old). The right side of Figure 3 displays this relation graphically. The REM model of the word-frequency effect described by Shiffrin and Steyvers [13, 18, 19] is a member of this class of models, as we describe next.

REM [18] assumes that memory traces consist of vectors $\underline{V}$, of length $\underline{w}$, of nonnegative integer feature values. Zero represents no information about a feature. Otherwise the values for a given feature are assumed to follow the geometric probability distribution given as Equation 1: $P(V = j) = (1-g)^{j-1}g$, for $j = 1$ and higher. Thus higher integer values represent feature values less likely to be encountered in the environment. REM adopts a "feature-frequency" assumption [13]: the lexical/semantic traces of lower frequency words are generated with a lower value of g (i.e. $g_{LF} < g_{HF}$). These lexical/semantic traces represent general knowledge (e.g., the orthographic, phonological, semantic, and contextual characteristics of a word) and have very many non-zero feature values, most of which are encoded correctly. Episodic traces represent the occurrence of stimuli in a certain environmental context; they are built of the same feature types as lexical/semantic traces, but tend to be incomplete (have many zero values) and inaccurate (the values do not necessarily represent correctly the values of the presented event).

When a word is studied, an incomplete and error prone representation of the word's lexical/semantic trace is stored in a separate episodic image. The probability that a feature will be stored in the episodic image after $\underline{t}$ time units of study is given as Equation 2: $1 - (1 - \underline{u}^*)^t$, where $\underline{u}^*$ is the probability of storing a feature in an arbitrary unit of time. The number of attempts, $t_j$, at storing a content feature for an item studied for $\underline{j}$ units of time is computed from Equation 3: $t_j = t_{j-1}(1 + \underline{e}^{-aj})$, where

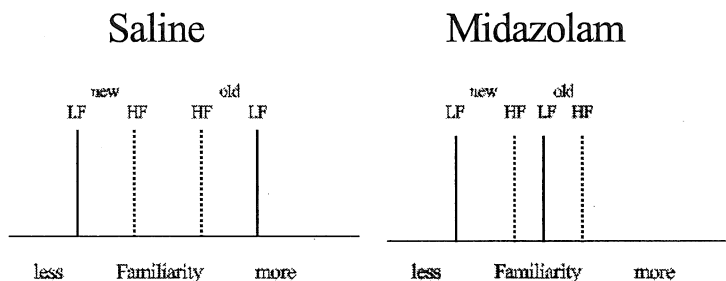

Figure 3. Arrangement of means of the theoretical distributions of strength-based models that may give rise to Hirshman et al.'s findings. HF and LF = high or LF frequency words, respectively.

$\underline{a}$ is a rate parameter, and $t_1$ is the number of attempts at storing a feature in the first 1 s. of study. Thus, increased study time increases the storage of features, but the gain in the amount of information stored diminishes as the item is studied longer. Features that are not copied from the lexical/semantic trace are represented by a value of 0. If storage of a feature does occur, the feature value is correctly copied from the word's lexical/semantic trace with probability $\underline{c}$. With probability $1-\underline{c}$ the value is incorrectly copied and sampled randomly from the long-run base-rate geometric distribution, a distribution defined by g such that $g_{HF} > g > g_{LF}$.

At test, a probe made with context features only is assumed to activate the episodic traces, $I_j$, of the $\underline{n}$ list items and no others [24]. Then the content features of the probe cue are matched in parallel to the activated traces. For each episodic trace, $\underline{I}_j$, the system notes the values of features of $\underline{I}_j$ that match the corresponding feature of the cue ($\underline{n}_{ijm}$ stands for the number of matching values in the j-th image that have value $\underline{i}$), and the number of mismatching features ($\underline{n}_{jq}$ stands for the number of mismatching values in the $j^{th}$ image). Next, a likelihood ratio, $\underline{\lambda}_j$, is computed for each $\underline{I}_j$:

$$\lambda_j = \left(1-c\right)^{n_{jq}} \prod_{i=1}^{\infty} \left[ \frac{c + (1-c)g(1-g)^{i-1}}{g(1-g)^{i-1}} \right]^{n_{ijm}} \qquad (4)$$

$\underline{\lambda}_j$ is the likelihood ratio for the $j^{th}$ image. It can be thought of as a match-strength between the retrieval cue and $\underline{I}_j$. It gives the probability of the data (the matches and mismatches) given that the retrieval cue and the image represent the same word (in which case features are expected to match, except for errors in storage) divided by the probability of the data given that the retrieval cue and the image represent different words (in which case features match only by chance).

The recognition decision is based on the odds, $\Phi$, giving the probability that the test item is old divided by the probability the test item is new [18]. This is just the average of the likelihood ratios:

$$\Phi = \frac{1}{n}\sum_{j=1}^{n} \lambda_j \qquad (5)$$

If the odds exceed a criterion, then an "old" response is made. The default criterion is 1.0 (which maximizes probability correct) although subjects could of course deviate from this setting.

Thus an "old" response is given when there is more evidence that the test word is old. Matching features contribute evidence that an item is old (contribute factors to the product in Eq. 3 greater than 1.0) and mismatching features contribute evidence that an item is new (contribute factors less than 1.0). REM predicts an effect of study time because storage of more non-zero features increases the number of matching target-trace features; this factor outweighs the general increase in variance produced by greater numbers of non-zero features in all vectors. REM predicts a LF HR advantage because the matching of the more uncommon features associated with LF words produces greater evidence that the item is old than the matching of the more common features associated with HF words. For foils, however, every feature match is due to chance; such matching occurs more frequently for HF than LF words because HF features are more common [12]. This factor outweighs the higher diagnosticity of matches for the LF words, and HF words are predicted to have higher FARs than LF words.

Much evidence points to the critical role of the hippocampal region in storing episodic memory traces [1, 14, 15, 16, 20]. Interestingly, Midazolam has been shown to affect the storage, but not the retrieval of memory traces [22]. As described above, there are two parameters in REM that affect the storage of features in memory: $\underline{u}^*$ determines the number of features that get stored, and $\underline{c}$ determines the accuracy with which features get stored. In order to lower performance, it could be assumed that Midazolam reduces the values of either or both of these parameters. However, Hirshman et al.'s data constrain which of these possibilities is viable.

Let us assume that Midazolam only causes the hippocampal region to store fewer features, relative to the saline condition (i.e. $u^*$ is reduced). In REM, this

causes fewer terms in the product given by Eq. 4, and a lower value for the result, on the average. Hence, if Midazolam causes fewer features to be stored, subjects should approach chance-level performance for both HF and LF words: LF(FAR) ~ HF(FAR) ~ LF(HR) ~ HF(HR). However, Hirshman et al. found that the difference in the LF and HF FARs was not affected by Midazolam. In REM this difference would not be much affected, if at all, by changes in criterion, or changes in $g$, that one might assume Midazolam induces. Thus within the framework of REM, the main effect of Midazolam on the functioning of the hippocampal region is not to reduce the number of features that get stored.

Alternatively let us assume that Midazolam causes the hippocampal region to store "noisier" episodic traces, as opposed to traces with fewer non-zero features, instantiated in REM by decreasing the value of the $c$ parameter (that governs correct copying of a feature value). Decreasing $c$ only slightly affects the false alarm rates, because these FARs are based on chance matches[1]. However, decreasing $c$ causes the LF and HF old-item distributions (see Figure 3) to approach the LF and HF new-item distributions; when the decrease is large enough, this factor must cause the LF and HF old-item distributions to reverse position. The reversal occurs because the HF retrieval cues used to probe memory have more common features (on average) than the LF retrieval cues, a factor that comes to dominate when the true 'signal' (matching features in the target trace) begins to disintegrate into noise (due to lowering of $c$).

Figure 1 shows predictions of a REM model incorporating the assumption that only $c$ varies between the saline and Midazolam groups, and only at storage. For retrieval the same $c$ value was used in both the saline and Midazolam conditions to calculate the likelihoods in Equation 4 (an assumption consistent with retrieval tuned to the participant's lifetime learning, and consistent with prior findings showing that Midazolam affects the storage of traces and not their retrieval [17]. The criterion for an old/new judgment was set to .92, rather than the normatively optimal value of 1.0, in order to obtain a good quantitative fit, but the criterion did not vary between the Midazolam and saline groups, and therefore is not of consequence for the present article. Within the REM framework, then, the main effect of Midazolam is to cause the hippocampal region to store more noisy episodic traces. These conclusions are based on the recognition data. We turn next to the remember/know judgments.

We chose to model remember-know judgments in what is probably the simplest way. The approach is based on the models described by Donaldson [4] and Hirshman and Master [10, 11]. As described above, an 'old' decision is given when the familiarity (i.e. activation, or in REM terms the odds) associated with a test word exceeds the yes-no criterion. When this happens, then it is assumed that a higher remember/know criterion is set. Words whose familiarity exceeds the higher remember/know criterion are given the "remember" response, and a "know" response is given when the remember/know criterion is not exceeded. Figure 2 shows that this model predicts the effects of Midazolam and saline both qualitatively and quantitatively. This fit was obtained by using slightly different remember-know criteria in the saline and Midazolam conditions (1.40 and 1.26 in the saline and Midazolam conditions, respectively), but all the qualitative effects are predicted correctly even when the same criterion is adopted for remember/know.

These predictions provide an existence proof that Hirshman et al. [9] were a bit hasty in using their data to reject single-process models of the present type [4, 11], and show that single- versus dual-process models would have to be distinguished on the basis of other sorts of studies. There is already a large literature devoted to this as-yet-unresolved issue [10], and space prevents discussion here.

Thus far we demonstrated the sufficiency of a model assuming that Midazolam reduces storage accuracy rather than storage quantity, and have argued that the reverse assumption cannot work. What degree of mixture of these assumptions might be compatible with the data? An answer would require an exhaustive exploration of the parameter space, but we found that the use of a 50% reduced value of $u^*$ for the Midazolam group ($u^*_{sal} = .02$; $u^*_{mid} = .01$) predicted an LF-FAR advantage that deviated from the data by being noticeably smaller in the Midazolam than saline condition. Within the REM framework this result suggests the main effect of Midazolam (possibly all the effect) is on $c$ (accuracy of storage) rather than on $u^*$ (quantity of storage).

Alternatively, it is possible to conceive of a much more complex REM model that assumes that the effect of Midazolam is to reduce the amount of storage. Accordingly, one might assume that relatively little information is stored in memory in the Midazolam condition, and that the retrieval cue is matched primarily against traces stored prior to the experiment. Such a model might predict Hirshman et al.'s findings because once again targets will only be randomly similar to contents of memory. However, such a model is far more complex than the model described above. Perhaps, future research will provide data that requires a more complex model, but for now the simple model presented here is sufficient.

## 4 Neuroscientific Speculations

The hippocampus (proper) consists of approximately 10% GABAergic interneurons, and these interneurons are thought to control the firing of the remaining 90% of the hippocampal principle neurons [21]. Some of the principle neurons are granule neurons and some are pyramidal neurons. The granule cells are associated with a rhythmic pattern of neuronal activity known as theta waves [1]. Theta waves are associated with exploratory activities in both animals [16] and humans [2], activities in which information about novel situations is being acquired. Midazolam is a benzodiazepine, and benzodiazepines inhibit the firing of GABAergic interneurons in the hippocampus [3]. Hence, if Midazolam inhibits the firing of those cells that regulate the orderly firing of the vast majority of hippocampal cells, then it is a reasonable to speculate that the result is a "noisier" episodic memory trace.

The argument that Midazolam causes noisier storage rather than less storage raises the question whether a similar process produces the similar effects caused by hippocampal lesions or other sorts of damage (e.g. Korsakoff's syndrome). This question could be explored in future research.

## Footnotes

[1] Slight differences are predicted depending on the interrelations of $g$, $g_{HF}$, and $g_{LF}$

## References

[1] Bazsaki, G. (1989). Two-stage model of memory trace formation: A role for "noisy" brain states. *Neuroscience*, 31, 551-570.

[2] Caplan, J. B., Raghavachari, S. and Madsen, J. R., Kahana, M. J. (2001). Distinct patterns of brain oscillations underlie two basic parameters of human maze learning. *J. of Neurophys.*, 86, 368-380.

[3] Deadwyler, S. A., West, M., & Lynch, G. (1979). Activity of dentate granule cells during learning: differentiation of perforant path input. *Brain Res.*, 169, 29-43.

[4] Donaldson, W. (1996). The role of decision processes in remembering and knowing. *Memory & Cognition, 24*, 523-533.

[5] Gardiner, J. M. (1988). Functional aspects or recollective experience. *Memory & Cognition, 16*, 309-313.

[6] Gillund, G., & Shiffrin, R. M. (1984). A retrieval model for both recognition and recall. *Psych. Rev.*, 91, 1-67.

[7] Glanzer, M., & Adams, J. K. (1985). The mirror effect in recognition memory. *Memory & Cognition*, 12, 8-20.

[8] Green, D. M., & Swets, J. A. (1966). *Signal detection theory and psychophysics*. New York: Wiley.

[9] Hirshman, E., Fisher, J., Henthorn, T., Arndt, J., & Passannante, A. (in press) Midazolam amnesia and dual-process models of the word frequency mirror effect. *J. of Memory and Language*, 47, 499-516.

[10] Hirshman, E. & Henzler, A. (1998) The role of decision processes in conscious memory, *Psych. Sci.*, 9, 61-64.

[11] Hirshman, E. & Master, S. (1997) Modeling the conscious correlates of recognition memory: Reflections on the Remember-Know paradigm. *Memory & Cognition*, 25, 345-352.

[12] Malmberg, K. J. & Murnane, K. (2002). List composition and the word-frequency effect for recognition memory. *J. of Exp. Psych.: Learning, Memory, and Cognition*, 28, 616–630.

[13] Malmberg, K. J., Steyvers, M., Stephens, J. D., & Shiffrin, R. M. (in press). Feature frequency effects in recognition memory. *Memory & Cognition*.

[14] Marr, D. (1971). Simple memory: a theory for the archicortex. *Proceedings of the Royal Society*, London B 841, 262:23-81.

[15] McClelland, J. L., McNaughton, B. L., & O'Reilly, R. C. (1995). Why there are complementary learning systems in the hippocampus and neocortex: Insights from the successes and failures of connectionist models of learning and memory. *Psych. Rev.*, 102, 419-457.

[16] O'Keefe, J. & Nadel, L. (1978). *The hippocampus as a cognitive map*. Oxford: Clarendon University Press.

[17] Polster, M., McCarthy, R., O'Sullivan, G., Gray, P., & Park, G. (1993). Midazolam-induced amnesia: Implications for the implicit/explicit memory distinction. *Brain & Cognition*, 22, 244-265.

[18] Shiffrin, R.M., & Steyvers, M. (1997). A model for recognition memory: REM – retrieving effectively from memory. *Psychonomic Bulletin & Review*, 4, 145-166.

[19] Shiffrin, R. M. & Steyvers, M. (1998). The effectiveness of retrieval from memory. In M. Oaksford & N. Chater (Eds.), *Rational models of cognition* (pp. 73-95). London: Oxford University Press.

[20] Squire, L. R. (1987). *Memory and the Brain*. New York: Oxford.

[21] Vizi, E. S. & Kiss, K. P. (1998). Neurochemistry and pharmacology of the major hippocampal transmitter systems: Synaptic and Nonsynaptic interactions. *Hippocampus*, 8, 566-607.
